# Delay Compensation with Dynamical Synapses

**C. C. Alan Fung, K. Y. Michael Wong**
Hong Kong University of Science and Technology, Hong Kong, China
alanfung@ust.hk,  phkywong@ust.hk

**Si Wu**
State Key Laboratory of Cognitive Neuroscience and Learning,
Beijing Normal University, Beijing 100875, China
wusi@bnu.edu.cn

## Abstract

Time delay is pervasive in neural information processing. To achieve real-time tracking, it is critical to compensate the transmission and processing delays in a neural system. In the present study we show that dynamical synapses with short-term depression can enhance the mobility of a continuous attractor network to the extent that the system tracks time-varying stimuli in a timely manner. The state of the network can either track the instantaneous position of a moving stimulus perfectly (with zero-lag) or lead it with an effectively constant time, in agreement with experiments on the head-direction systems in rodents. The parameter regions for delayed, perfect and anticipative tracking correspond to network states that are static, ready-to-move and spontaneously moving, respectively, demonstrating the strong correlation between tracking performance and the intrinsic dynamics of the network. We also find that when the speed of the stimulus coincides with the natural speed of the network state, the delay becomes effectively independent of the stimulus amplitude.

## 1 Introduction

Time delay is pervasive in neural information processing. Its occurrence is due to the time for signals to transmit in the neural pathways, e.g., 50-80 ms for electrical signals to propagate from the retina to the primary visual cortex [13], and the time for neurons responding to inputs, which is in the order of 10-20 ms. Delay is also inevitable for neural information processing. For a neural system carrying out computations in the temporal domain, such as speech recognition and motor control, input information needs to be integrated over time, which necessarily incur delays.

To achieve real-time tracking of fast moving objects, it is critical for a neural system to compensate for the delay; otherwise, the object position perceived by the neural system will lag behind the true object position considerably. A natural way to compensate for delays is to predict the future position of the moving stimulus. Experimental findings suggested that delay compensations are widely adopted in neural systems. A remarkable example is the head-direction (HD) systems in rodents, which encode the head direction of a rodent in the horizontal plane relative to a static environment [14, 17]. It was found that when the head of a rodent is moving continuously in space, the direction perceived by the HD neurons in the postsubicular cortex has nearly zero-lag with respect to the instantaneous position of the rodent head [18]. More interestingly, in the anterior dorsal thalamic nucleus, the HD neurons perceive the future direction of the rodent head, leading the current position by a constant time [3]. The similar anticipative behavior is also observed in the eye-position neurons when animals make saccadic eye movement, the so-called saccadic remapping [16]. In human psychophysical experiments, the classic flash-lag effect also supports the notion of delay

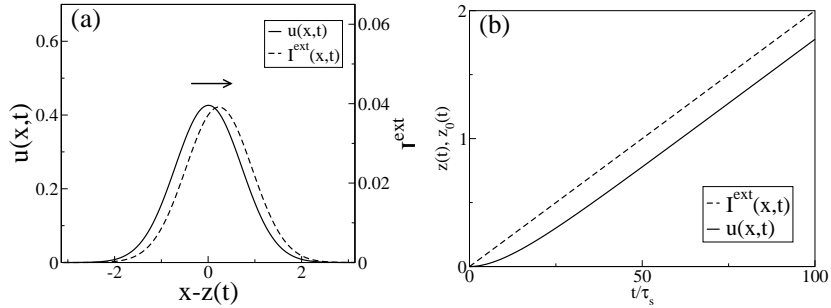

Figure 1: (a) Profiles of $u(x,t)$ and $I^{\text{ext}}(x,t)$ in the absence of STD, where the center of mass of the stimulus is moving with constant velocity $v = 0.02a/\tau_s$. As shown, the profile of $u(x,t)$ is almost Gaussian. (b) The centers of mass of $u(x,t)$ and $I^{\text{ext}}(x,t)$ as functions of time. Parameters: $\rho = 128/2\pi$, $a = 0.5$, $J_0 = \sqrt{2\pi}a$ and $\rho J_0 A = 1.0$.

compensation [12]. In the experiment, a flash is perceived to lag behind a moving object, even though they are physically aligned. The underlying cause is that the visual system predicts the future position of the continuously moving object, but is unable do so for the unpredictable flash.

Depending on the available information, the brain may employ different strategies for delay compensation. In the case of self-motion, such as an animal rotating its head actively or performing saccadic eye movements, the motor command responsible for the motion can serve as a cue for delay compensation. It was suggested that an efference copy of the motor command, called corollary discharge, is sent to the corresponding internal representation system prior to the motion [18]. For the head rotation, the advanced time can be up to 20 ms; for the saccadic eye movement, the advanced time is about 70 ms. In the case of tracking an external moving stimulus, the neural system has to rely on the moving speed of the stimulus for prediction. Asymmetric neural interactions have been proposed to drive the network states to catch up with changes in head directions [22] or positions [4]. These may be achieved by the so-called conjunctive cells projecting neural signals between successive modules in forward directions [10]. To explain the flash-lag effect, Nijhawan et al. proposed a dynamical routing mechanism to compensate the transmission delay in the visual system, in which retinal neurons dynamically choose a pathway according to the speed of the stimulus, and transmit the signal directly to the future position in the cortex [13].

In this study we propose a novel mechanism of how a neural system compensates for the processing delay. By the processing delay, we mean the time consumed by a neural system in response to external inputs. The proposed mechanism does not require corollary discharge, or efforts of choosing signal pathways, or specific network structures such as asymmetric interactions or conjunctive cells. It is based on the short-term depression (STD) of synapses, the inherent and ubiquitous nature that the synaptic efficacy of a neuron is reduced after firing due to the depletion of neurotransmitters [11]. It has been found that STD enhances the mobility of the states of neural networks [21, 9, 6]. The underlying mechanism is that neurotransmitters become depleted in the active region of the network states compared with the neighboring regions, thus increasing the likelihood of the locally active network state to shift to its neighboring positions when it is tracking a continuously shifting stimulus. When STD is sufficiently strong, the tracking state of the network can even overtake the moving stimulus, demonstrating its potential for generating predictions.

## 2 The Model

We consider continuous attractor neural networks (CANNs) as the internal representation models for continuous stimuli [7, 2, 15]. A CANN holds a continuous family of bump-shaped stationary states, which form a subspace in which the neural system is neutrally stable [20]. This property endows the neural system the capacity of tracking time-varying stimuli smoothly.

Consider a continuous stimulus $x$ being encoded by a neural ensemble. The variable $x$ may represent the orientation, the head direction, or the spatial location of an object. Neurons with preferred stimuli $x$ produce the maximum response when an external stimulus is present at $x$. Their preferred stimuli

are uniformly distributed in the space $-\infty < x < \infty$. In the continuum limit, the dynamics of the neural ensemble can be described by a CANN. We denote as $u(x,t)$ the population-averaged synaptic current to the neurons at position $x$ and time $t$. The dynamics of $u(x,t)$ is determined by the external input, the lateral interactions among the neurons, and its relaxation towards zero response. It is given by

$$\tau_s \frac{\partial u(x,t)}{\partial t} = I^{\text{ext}}(x,t) + \rho \int dx' J(x,x') p(x',t) r(x',t) - u(x,t), \tag{1}$$

where $\tau_s$ is the synaptic time constant, which is typically in the order of 1 to 5 ms, $I^{\text{ext}}(x,t)$ the external input, $\rho$ the density of neurons, $J(x,x')$ the coupling between neurons at $x$ and $x'$, and $r(x,t)$ is the firing rate of the neurons. The variable $p(x,t)$ represents the fraction of available neurotransmitters, which evolves according to [6, 19]

$$\tau_d \frac{\partial p(x,t)}{\partial t} = 1 - p(x,t) - \tau_d \beta p(x,t) r(x,t), \tag{2}$$

where $\tau_d$ is the STD time scale, which is typically of the order of $10^2$ ms. In this work, we choose $\tau_d = 50\tau_s$. The STD effect is controlled by the parameter $\beta$, which can be considered as the fraction of total neurotransmitters consumed per spike.

The actual forms of $J(x,x')$ and $r(x,t)$ depend on the details of the neural dynamics. Here, for the convenience of analysis, we choose them to be

$$J(x,x') = \frac{J_0}{a\sqrt{2\pi}} \exp\left[-\frac{(x-x')^2}{2a^2}\right], \tag{3}$$

$$r(x,t) = \Theta[u(x,t)] \frac{u(x,t)^2}{1 + k\rho \int dx' u(x',t)^2}, \tag{4}$$

where $J_0$ and $a$ control the magnitude and range of the neuronal excitatory interactions respectively. $J(x,x')$ is translationally invariant in the space $x$, since it is a function of $(x-x')$, which is essential for the network state to be neutrally stable. In the expression for the firing rate, $\Theta$ is the step function. Here, the stabilizing effect of inhibitory interactions is achieved by the divisive normalization operation in Eq. (4).

Let us consider first the case without STD by setting $\beta = 0$. Hence, $p(x,t) = 1$ in Eq. (1). For $k \leq k_c \equiv \rho J_0^2/(8\sqrt{2\pi}a)$, the network holds a continuous family of Gaussian-shaped stationary states when $I^{ext}(x,t) = 0$. These stationary states are

$$\bar{u}(x) = \bar{u}_0 \exp\left[-\frac{(x-z)^2}{4a^2}\right]. \tag{5}$$

where $\bar{u}$ is the rescaled variable $\bar{u} \equiv \rho J_0 u$, and $\bar{u}_0$ is the rescaled bump height. The parameter $z$, i.e., the center of the bump, is a free parameter, implying that the stationary state of the network can be located anywhere in the space $x$.

Next, we consider the case that the network receives a moving input,

$$I^{\text{ext}}(x,t) = A \exp\left[-\frac{(x-z_0(t))^2}{4a^2}\right], \tag{6}$$

where $A$ is the magnitude of the input and $z_0$ the stimulus position.

Without loss of generality, we consider the stimulus position at time $t = 0$ to be $z_0 = 0$, and the stimulus moves at a constant speed thereafter, i.e., $z_0 = vt$ for $t \geq 0$. Let $s \equiv z(t) - z_0(t)$ be the displacement between the network state and the stimulus position. It has been shown that without STD, the steady value of the displacement is determined by [5]

$$v = -\frac{As}{\tau_s} \exp\left(-\frac{s^2}{8a^2}\right). \tag{7}$$

Note that $s$ has the opposite sign of $v$, implying that the network state always trails behind the stimulus (see Fig. 1(a)). This is due to the response delay of the network relative to the input.

# 3    Tracking in the Presence of STD

The analysis of tracking in the presence of STD is more involved. Motivated by the nearly Gaussian-shaped profile of the network states, we adopt a perturbation approach to solve the network dynamics [5]. The key idea is to expand the network states as linear combinations of a set of orthonormal basis functions corresponding to different distortion modes of the bump, that is,

$$u(x,t) = \sum_n u_n(t)\psi_n(x-z),\tag{8}$$

$$1 - p(x,t) = \sum_n p_n(t)\phi_n(x-z),\tag{9}$$

where the basis functions are

$$\psi_n(x-z) = \frac{1}{\sqrt{\sqrt{2\pi}a2^n n!}}H_n\left(\frac{x-z}{\sqrt{2}a}\right)\exp\left[-\frac{(x-z)^2}{4a^2}\right],\tag{10}$$

$$\phi_n(x-z) = \frac{1}{\sqrt{\sqrt{\pi}a2^n n!}}H_n\left(\frac{x-z}{a}\right)\exp\left[-\frac{(x-z)^2}{2a^2}\right].\tag{11}$$

Here, $H_n$ is the $n^{\text{th}}$-order Hermite polynomial function. $\psi_n(x-z)$ and $\phi_n(x-z)$ have clear physical meanings. For instance, for $n = 1, 2, 3, 4$, they corresponds to, respectively, the height, the position, the width and the skewness changes of the Gaussian bump. Depending on the approximation precision, we can take the above expansions up to a proper order, and substituting them into Eqs. (1) and (2) to solve the network dynamics analytically.

Results obtained from the $11^{th}$ order perturbation are shown in Fig. 2(a) for three representative cases. They depend on the rescaled inhibition $\bar{k} \equiv k/k_c$ and the rescaled STD strength $\bar{\beta} \equiv \tau_d\beta/(\rho^2 J_0^2)$. When STD is weak, the tracking state lags behind the stimulus. When the STD strength increases to a critical value $\bar{\beta}_{\text{perfect}}$, $s$ becomes effectively zero in a rather broad range of stimulus velocity, achieving *perfect tracking*. When the STD strength is above the critical value, the tracking state leads the stimulus.

Hence delay compensation in a tracking task can be implemented at two different levels. The first one is *perfect tracking*, in which the tracking state has zero-lag with respect to the true stimulus position independent of the stimulus speed. The second one is *anticipative tracking*, in which the tracking state leads by a constant time $\tau_{\text{ant}}$ relative to the stimulus position, that is, the tracking state is at the position the stimulus will travel to at a later time $\tau_{\text{ant}}$. To achieve a constant anticipation time, it requires the leading displacement to increase with the stimulus velocity proportionally, i.e., $s = v\tau_{\text{ant}}$. Both forms of delay compensation have been observed in the head-direction systems of rodents, and may serve different functional purposes.

## 3.1    Prefect Tracking

To analyze the parameter regime for perfect tracking, it is instructive to consider the $1^{\text{st}}$ order perturbation of the network dynamics, i.e.,

$$u[x - z(t)] = u_0(t)\exp\left[-\frac{(x-z(t))^2}{4a^2}\right],\tag{12}$$

$$p[x - z(t)] = 1 - p_0(t)\exp\left[-\frac{(x-z(t))^2}{2a^2}\right] + p_1(t)\left[\frac{x-z(t)}{a}\right]\exp\left[-\frac{(x-z(t))^2}{2a^2}\right].\tag{13}$$

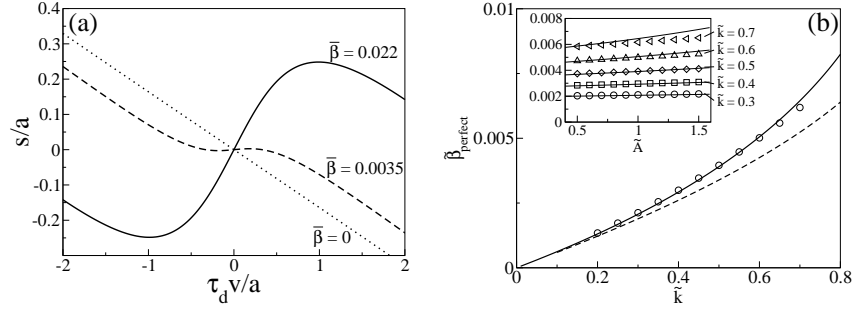

Figure 2: (a) The dependence of the displacement between the bump and the stimulus on the velocity of the moving stimulus for different values of $\bar{\beta}$. Parameters: $\bar{k} = 0.4$ and $\bar{A} = 1.8$. (b) The dependence of $\bar{\beta}_{\text{perfect}}$ on $\bar{k}$ with $\bar{A} = 1.0$. Symbols: simulations. Solid line: the predicted curve of $\bar{\beta}_{\text{perfect}}$. Dashed line: the boundary separating the static and metastatic phases according to the $1^{\text{st}}$ order perturbation [6]. Inset: the dependence of $\bar{\beta}_{\text{perfect}}$ on $\bar{A}$. Symbols: simulations. Lines: theoretical prediction according to the $1^{\text{st}}$ order perturbation.

Substituting them into Eqs. (1) and (2) and utilizing the orthogonality of the basis functions, we get (see Supplementary Material)

$$\tau_s \frac{d\bar{u}_0}{dt} = \frac{\bar{u}_0^2}{B\sqrt{2}}\left(1 - p_0\sqrt{\frac{4}{7}}\right) - \bar{u}_0 + \bar{A}e^{-\frac{(vt-z)^2}{8a^2}}, \tag{14}$$

$$\frac{\tau_s}{2a}\frac{dz}{dt} = \frac{\bar{u}_0}{B}\left(\frac{2}{7}\right)^{3/2}p_1 + \frac{\bar{A}}{2\bar{u}_0}\left(\frac{vt-z}{a}\right)e^{-\frac{(vt-z)^2}{8a^2}}, \tag{15}$$

$$\tau_s \frac{dp_0}{dt} = \frac{\tau_s}{\tau_d}\left[\frac{\bar{\beta}\bar{u}_0^2}{B}\left(1 - p_0\sqrt{\frac{2}{3}}\right) - p_0\right] - \frac{\tau_s p_1}{2a}\frac{dz}{dt}, \tag{16}$$

$$\frac{\tau_s}{p_0}\frac{dp_1}{dt} = -\frac{\tau_s}{\tau_d}\left[1 + \frac{\bar{\beta}\bar{u}_0^2}{B}\left(\frac{2}{3}\right)^{3/2}\right]\frac{p_1}{p_0} + \frac{\tau_s}{a}\frac{dz}{dt}. \tag{17}$$

At the steady state, $d\bar{u}_0/dt = dp_0/dt = dp_1/dt = 0$, and $dz/dt = v$. Furthermore, for a sufficiently small displacements, i.e., $|s|/a \ll 1$, one can approximate $\bar{A}\exp[-(vt-z)^2/(8a^2)] \approx \bar{A}$ and $\bar{A}[(vt-z)/a]\exp[-(vt-z)^2/(8a^2)] \approx -\bar{A}s/a$. Solving the above equations, we find that $s/a$ can be expressed in terms of the variables $\bar{u}_0/\bar{A}$, $\tau_s/\tau_d$ and $v\tau_d/a$. When $v\tau_d/a \ll 1$, the rescaled displacement $s/a$ can be approximated by a power series expansion of the rescaled velocity $v\tau_d/a$. Since the displacement reverses sign when the velocity reverses, $s/a$ is an odd function of $v\tau_d/a$. This means that $s/a \approx c_1(v\tau_d/a) + c_3(v\tau_d/a)^3$. For perfect tracking in the low velocity limit, we have $c_1 = 0$ and find

$$\frac{s}{a} = -\frac{C}{2}\frac{\bar{u}_0}{\bar{A}}\frac{\tau_s}{\tau_d}\left(\frac{v\tau_d}{a}\right)^3, \tag{18}$$

where $C$ is a parameter less than 1 (the detailed expression can be found in Supplementary Material). For the network tracking a moving stimulus, the input magnitude cannot be too small. This means that $\bar{u}_0/\bar{A}$ is not a large number. Therefore, for tracking speeds up to $v\tau_d/a \sim 1$, the displacement $s$ is very small and can be regarded as zero effectively (see Fig. 2(a)). The velocity range in which the tracking is effectively perfect is rather broad, since it scales as $(\tau_d/\tau_s)^{1/3} \gg 1$.

Equation (18) is valid when $\bar{\beta}$ takes a particular value. This ields an extimate of $\bar{\beta}_{\text{perfect}}$ in the $1^{\text{st}}$ order perturbation. Its expression is derived in Supplementary Material and plotted in Fig. 2(b). For reference, we also plot the boundary that separates the metastatic phase above it from the static phase below, as reported in the study of intrinsic properties of CANNs with STD in [6]. In the static phase, the bump is stable at any position, whereas in the metastatic phase, the static bump starts to move spontanaeously once it is pushed. Hence we say that the phase boundary is in a *ready-to-move* state. Fig. 2(b) shows that $\bar{\beta}_{\text{perfect}}$ is just above the phase boundary. Indeed, when $\bar{A}$ approaches 0, the expression of $\bar{\beta}_{\text{perfect}}$ reduces to the value of $\bar{\beta}$ along the phase boundary for the $1^{\text{st}}$ order

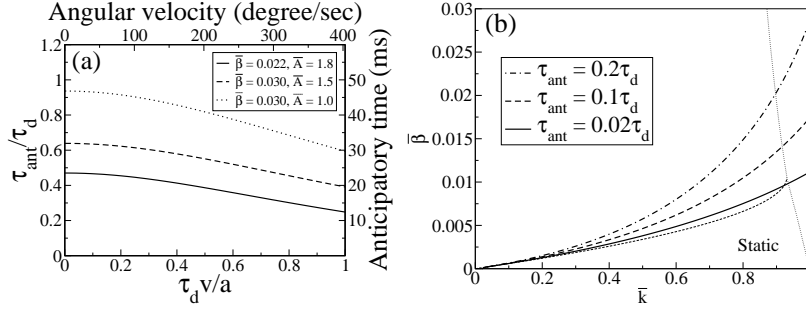

Figure 3: (a) The anticipatory time as a function of the speed of the stimulus. Different sets of parameters may correspond to different levels of anticipatory behavior. Parameter: $\bar{k} = 0.4$. The numerical scales are estimated from parameters in [8]. (b) The contours of constant anticipatory time in the space of rescaled inhibition $\bar{k}$ and the rescaled STD strength $\bar{\beta}$ in the limit of very small stimulus speed. Dashed line: boundary separating the static and metastatic phases. Dotted line: boundary separating the existence and non-existence phases of bumps. Calculations are done using $11^{\text{th}}$ order perturbation.

perturbation. The inset of Fig. 2(b)) confirms that $\bar{\beta}_{\text{perfect}}$ does not change significantly with $\bar{A}$ for different values of $\bar{k}$. This implies that the network with $\bar{\beta} = \bar{\beta}_{\text{perfect}}$ exhibits effectively perfect tracking performance because it is intrinsically in a ready-to-move state.

## 3.2 Anticipative Tracking

We further explore the network dynamics when the STD strength is higher than that for achieving perfect tracking. By solving the network dynamics with the perturbation expansion up to the $11^{\text{th}}$ order, we obtain the relation between the displacement $s$ and the stimulus speed $v$. The solid curve in Fig. 2(a) shows that for strong STD, $s$ increases linearly with $v$ over a broad range of $v$. This implies that the network achieves a constant anticipatory time $\tau_{\text{ant}}$ over a broad range of the stimulus speed.

To gain insights into how the anticipation time depends on the stimulus speed, we consider the regime of small displacements. In this regime, the rescaled displacement $s/a$ can be approximated by a power series expansion of the rescaled velocity $v\tau_d/a$, leading to $s/a = c_1(v\tau_d/a) + c_3(v\tau_d/a)^3$. The coefficients $c_1$ and $c_3$ are determined such that the anticipation time in the limit $v = 0$ should be $\tau_{\text{ant}}(0) = s/v$, and that $s/a$ reaches a maximum when $v = v_{\text{max}}$. This yields the result

$$\frac{s}{a} = \frac{\tau_{\text{ant}}(0)}{\tau_d}\left[\frac{v\tau_d}{a} - \frac{1}{3}\left(\frac{a}{v_{\text{max}}\tau_d}\right)^2\left(\frac{v\tau_d}{a}\right)^3\right].\tag{19}$$

Hence the anticipatory time is given by

$$\tau_{\text{ant}}(v) = \tau_{\text{ant}}(0)\left(1 - \frac{v^2}{3v_{\text{max}}^2}\right).\tag{20}$$

This shows that the anticipation time is effectively constant in a wide range of stimulus velocities, as shown in Fig. 3(a). Even for $v = 0.5v_{\text{max}}$, the anticipation time is only reduced from its maximum by 9%.

The contours of anticipatory times for slowly moving stimuli are shown in Fig. 3(b). Hence the region of anticipative behavior effectively coincides with the metastatic phase, as indicated by the region above the phase line (dashed) in Fig. 2(b). In summary, there is a direct correspondence between delayed, perfect, and anticipative tracking on one hand, and the static, ready-to-move, and spontaneously moving beahviors on the other. This demonstrates the strong correlation between the tracking performance and the intrinsic behaviors of the CANN.

We compare the prediction of the model with experimental data. In a typical HD experiment of rodents [8], $\tau_s = 1$ ms, $a = 28.5$ degree$/\sqrt{2}$, and the anticipation time drops from 20 ms at $v = 0$ to 15 ms at $v = 360$ degree/s. Substituting into Eq. (19) and assuming $\tau_d = 50\tau_s$, these parameters yield a slope of 0.41 at the origin and the maximum lead at $v_{\text{max}}\tau_d/a = 1.03$. This result can be

compared favorably with the curve of $\bar{\beta} = 0.022$ in Fig. 2(a), where the slope at the origin is $0.45$ and the maximum lead is located at $v_{\max}\tau_d/a = 1.01$. Based on these parameters, the lowest curve plotted in Fig. 3(a) is consistent with the real data in Fig. 4 of [8].

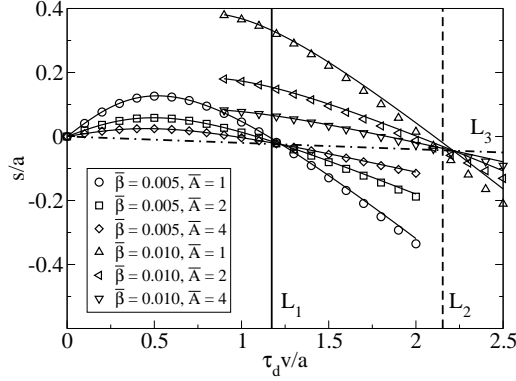

Figure 4: Confluence points at natural speeds. There are six curves in two groups with different sets of parameters. Curves in one group intersect at the confluence point with the natural speed at the corresponding value of $\bar{\beta}$. Symbols: simulations. Thin lines: prediction of the displacement-velocity relation by 11$^{\text{th}}$ order perturbation. $L_1$: natural speed at $\bar{\beta} = 0.005$. $L_2$: natural speed at $\bar{\beta} = 0.01$. $L_3$: the line for natural tracking at high $\bar{A}$ limit. Parameter: $\bar{k} = 0.3$.

## 3.3 Natural Tracking

For strong enough STD, a CANN holds spontaneously moving bump states. The speed of the spontaneously moving bump is an intrinsic property of the network depending only on the network parameters. We call this the natural speed of the network, denoted as $v_{\text{natural}}$. An interesting issue is the tracking performance of the network when the stimulus is moving at its natural speed.

Two sets of curves corresponding to two values of $\bar{\beta}$ are shown in Fig. 4, when the stimulus amplitude $\bar{A}$ is sufficiently strong. The lines $L_1$ and $L_2$ indicate the corresponding natural speeds of the system for these values of $\bar{\beta}$. Remarkably, we obtain a confluence point of these curves at the natural speed. This point is referred to as the *natural tracking point*. It has the important property that the lag is independent of the stimulus amplitude. This independence of $s$ from $\bar{A}$ persists in the asymptotic limit of large $\bar{A}$. In this limit, $s$ approaches $-v_{\text{natural}}\tau_s$ , corresponding to a delay time of $\tau_s$, showing that the response is limited by the synaptic time scale in this limit. This asymptotic limit is described by the line $L_3$ and is identical for all values of $\bar{k}$ and $\bar{\beta}$. Hence the invariant point for natural tracking is given by $(v, s) = (v_{\text{natural}}, -v_{\text{natural}}\tau_s)$ for all values of $\bar{k}$ and $\bar{\beta}$.

We also consider natural tracking in the weak $\bar{A}$ limit. Again we find a confluence point of the displacement curves at the natural speed, but the delay time (and in some cases the anticipation time) depends on the value of $\bar{k}$. For example, at $\bar{k} = 0.3$, the natural tracking point traces out an effectively linear curve in the space of $v$ and $s$ when $\bar{\beta}$ increases, with a slope equal to $0.8\tau_s$. This shows that the delay time is $0.8\tau_s$, effectively independent of $\bar{\beta}$ at $\bar{k} = 0.3$. Since the delay time is different from the value of $\tau_s$ applicable in the strong $\bar{A}$ limit, the natural tracking point is slowly drifting from the weak to the strong $\bar{A}$ limit. However, the magnitude of the natural time delay remains of the order of $\tau_s$. This is confirmed by the analysis of the dynamical equations when the stimulus speed is $v_{\text{natural}} + \delta v$ in the weak $\bar{A}$ limit.

## 3.4 Extension to other CANNs

To investigate whether the delay compensation behavior and the prediction of the natural tracking point are general features of CANN models, we consider a network with Mexican-hat couplings. We replace $J(x, x')$ in Eq. (1) by

$$J^{\text{MH}}(x, x') = J_0 \left[ \frac{1}{2} - \left( \frac{x - x'}{2a} \right)^2 \right] \exp\left[ -\frac{(x - x')^2}{2a^2} \right],$$ (21)

and $r(x, t)$ in Eqs. (1) and (2) by

$$r(x, t) = \Theta\left[u(x, t)\right] \frac{u(x, t)^2}{1 + u(x, t)^2}$$ (22)

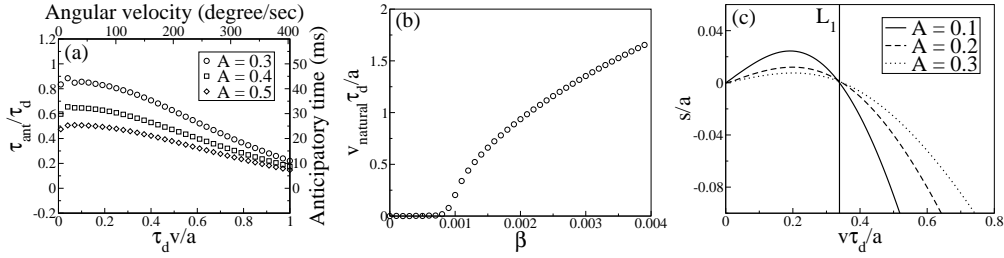

Figure 5: (a) The dependence of anticipatory time on the stimulus speed in the Mexican-hat model. Parameter: $\beta = 0.003$. (b) Natural speed of the network as a function of $\beta$. (c) Plot of $s$ against $v$. There is a confluence point at the natural speed of the system. $L_1$: the natural speed of the system at $\beta = 0.0011$. Common parameters: $\rho = 128/(2\pi)$, $J_0 = 0.5$ and $a = 0.5$.

Fig. 5 shows that the network exhibits the same behaviors as the model in Eqs. (1) and (2). As shown in Fig. 5(a), the anticipatory times are effectively constant and similar in magnitude in the range of stimulus speed comparable to experimental settings. In Fig. 5(b), the natural speed of the bump is zero for $\beta$ less than a critical value. As $\beta$ increases, the natural speed increases from zero. In Fig. 5(c), the displacement $s$ is plotted as a function of the stimulus speed $v$. The invariance of the displacement at the natural speed, independent of the stimulus amplitude, also appears in the Mexican-hat model. The confluence point of the family of curves is close to the natural speed. Furthermore, the displacement at the natural tracking point increases with the natural speed.

## 4    Conclusions

In the present study we have investigated a simple mechanism of how processing delays can be compensated in neural information processing. The mechanism is based on the intrinsic dynamics of a neural circuit, utilizing the STD property of neuronal synapses. The latter induces translational instability of neural activities in a CANN and enhances the mobility of the network states in response to external inputs. We found that for strong STD, the neural system can track moving stimuli with either zero-lag or a lead of a constant time. The conditions for perfect and anticipative tracking hold for a wide range of stimulus speeds, making them applicable in practice. By choosing biologically plausible parameters, our model successfully justifies the experimentally observed delay compensation behaviors. We also made an interesting prediction in the network dynamics, that is, when the speed of the stimulus coincides with the natural speed of the network state, the delay becomes effectively independent of the stimulus amplitude. We also studied more than one kind of CANN models to confirm the generality of our results.

Compared with other delay compensation strategies relying on corollary discharge or dynamical routing, the mechanism we propose here is fully dependent on the intrinsic dynamics of the network, namely, the network automatically "adjusts" its tracking speed according to the input information. There exists strong correlations between tracking performance and the intrinsic dynamics of the network. The parameter regions for delayed, perfect and anticipative tracking correspond to network states being static, ready-to-move and spontaneously moving, respectively. It has been suggested the anticipative response of HD neurons in anterior dorsal thalamus is due to the corollary discharge of motor neurons responsible for moving the head. However, experimental studies revealed that when rats were moved passively (and hence no corollary discharge is available), either by hand or by a chart, the anticipative response of HD neurons still exists and has an even larger leading time [1]. Our model provides a possible mechanism to describe this phenomenon.

**Acknowledgement**

This work is supported by the Research Grants Council of Hong Kong (grant number 605010) and the National Foundation of Natural Science of China (No.91132702, No.31221003).

# References

[1] J. P. Bassett, M. B. Zugaro, G. M. Muir, E. J. Golob, R. U. Muller and J. S. Taube. Passive Movements of the Head Do Not Abolish Anticipatory Firing Properties of Head Direction Cells. J. Neurophysiol. 93, 1304-1316 (2005).

[2] R. Ben-Yishai, R. Lev. Bar-Or, and H. Sompolinsky. Theory of orientation tuning in visual cortex. Proc. Natl. Acad. Sci. U.S.A. 92, 3844-3848 (1995).

[3] H. T. Blair and P. E. Sharp. Anticipatory head direction signals in anterior thalamus: evidence for a thalamocortical circuit that integrates angular head motion to compute head direction. J. Neurosci. 15, 6260-6270 (1995).

[4] M. C. Fuhs and D. S. Touretzky. J. Neurosci. 26, A Spin Glass Model of Path Integration in Rat Medial Entorhinal Cortex . 4266-4276 (2006).

[5] C. C. A. Fung, K. Y. Wong and S. Wu. Neural Comput. Moving Bump in a Continuous Manifold: A Comprehensive Study of the Tracking Dynamics of Continuous Attractor Neural Networks. 22, 752-792 (2010).

[6] C. C. A. Fung, K. Y. M. Wong, H. Wang and S. Wu. Dynamical Synapses Enhance Neural Information Processing: Gracefulness, Accuracy and Mobility. Neural Comput. 24, 1147-1185 (2012).

[7] A. P. Georgopoulos, J. T. Lurito, M. Petrides, Mental rotation of the neuronal population vector. A. B. Schwartz, and J. T. Massey, Science 243, 234-236 (1989).

[8] J. P. Goodridge and D. S. Touretzky. Modeling attractor deformation in the rodent head direction system. J. Neurophysiol.83, 3402-3410 (2000).

[9] Z. P. Kilpatrick and P. C. Bressloff. Effects of synaptic depression and adaptation on spatiotemporal dynamics of an excitatory neuronal network. Physica D 239, 547-560 (2010).

[10] B. L. McNaughton, F. P. Battaglia, O. Jensen, E. I. Moser and M.-B. Moser. Path integration and the neural basis of the 'cognitive map'. Nature Rev. Neurosci. 7, 663-678 (2006).

[11] H. Markram and M. Tsodyks. Redistribution of synaptic efficacy between neocortical pyramidal neurons. Nature 382, 807-810, 1996.

[12] R. Nijhawan. Motion extrapolation in catching. Nature 370, 256-257 (1994).

[13] R. Nijhawan and S. Wu. Phil. Compensating time delays with neural predictions: are predictions sensory or motor? Trans. R. Soc. A 367, 1063-1078 (2009).

[14] J. O'Keefe and J. Dostrovsky. The hippocampus as a spatial map: preliminary evidence from unit activity in the freely moving rat. Brain Res. 34, 171-175 (1971).

[15] A. Samsonovich and B. L. McNaughton. Path integration and cognitive mappping in a continuous attractor neural network model. J. Neurosci. 17, 5900-5920 (1997).

[16] M. A. Sommer and R. H. Wurtz. Influence of the thalamus on spatial visual processing in frontal cortex. Nature 444, 374-377 (2006).

[17] J. S. Taube, R. U. Muller and J. B. Ranck Jr. Head-direction cells recorded from the postsubiculum in freely moving rats. I. Description and quantitative analysis. J. Neurosci. 10, 420-435 (1990).

[18] J. S. Taube and R. I. Muller. Comparisons of head direction cell activity in the postsubiculum and anterior thalamus of freely moving rats. Hippocampus 8, 87-108 (1998).

[19] M. Tsodyks, K. Pawelzik, and H. Markram. Neural Networks with Dynamic Synapses. Neural Comput. 10, 821-835 (1998).

[20] S. Wu and S. Amari. Computing with Continuous Attractors: Stability and Online Aspects no access. Neural Comput. 17, 2215-2239 (2005).

[21] L. C. York and M. C. W. van Rossum. Recurrent networks with short term synaptic depression. J. Comput. Neurosci 27, 707-620 (2009).

[22] K. Zhang. Representation of spatial orientation bythe intrinsic dynamics of the head-direction cell ensemble: a theory. J. Neurosci. 16, 2112-2126 (1996).

